# b-Bit Minwise Hashing for Estimating Three-Way Similarities

**Ping Li**
Dept. of Statistical Science
Cornell University

**Arnd Christian König**
Microsoft Research
Microsoft Corporation

**Wenhao Gui**
Dept. of Statistical Science
Cornell University

## Abstract

Computing[1] two-way and multi-way set similarities is a fundamental problem. This study focuses on estimating 3-way **resemblance** (Jaccard similarity) using $b$-bit minwise hashing. While traditional minwise hashing methods store each hashed value using 64 bits, $b$-bit minwise hashing only stores the lowest $b$ bits (where $b \geq 2$ for 3-way). The extension to 3-way similarity from the prior work on 2-way similarity is technically non-trivial. We develop the precise estimator which is accurate and very complicated; and we recommend a much simplified estimator suitable for sparse data. Our analysis shows that $b$-bit minwise hashing can normally achieve a 10 to 25-fold improvement in the storage space required for a given estimator accuracy of the 3-way resemblance.

## 1   Introduction

The efficient computation of the similarity (or overlap) between sets is a central operation in a variety of applications, such as *word associations* (e.g., [13]), *data cleaning* (e.g., [40, 9]), *data mining* (e.g., [14]), *selectivity estimation* (e.g., [30]) or *duplicate document detection* [3, 4]. In machine learning applications, binary (0/1) vectors can be naturally viewed as sets. For scenarios where the underlying data size is sufficiently large to make storing them (in main memory) or processing them in their entirety impractical, probabilistic techniques have been proposed for this task.

**Word associations** (collocations, co-occurrences)    If one inputs a query *NIPS machine learning*, all major search engines will report the number of pagehits (e.g., one reports 829,003), in addition to the top ranked URLs. Although no search engines have revealed how they estimate the numbers of pagehits, one natural approach is to treat this as a set intersection estimation problem. Each word can be represented as a set of document IDs; and each set belongs to a very large space $\Omega$. It is expected that $|\Omega| > 10^{10}$. Word associations have many other applications in Computational Linguistics [13, 38], and were recently used for Web search query reformulation and query suggestions [42, 12].

Here is another example. Commercial search engines display various form of "vertical" content (e.g., *images, news, products*) as part of Web search. In order to determine from which "vertical" to display information, there exist various techniques to select verticals. Some of these (e.g., [29, 15]) use the number of documents the words in a search query occur in for different text corpora representing various verticals as features. Because this selection is invoked for all search queries (and the tight latency bounds for search), the computation of these features has to be very fast. Moreover, the accuracy of vertical selection depends on the number/size of document corpora that can be processed within the allotted time [29], i.e., the processing speed can directly impact quality.

Now, because of the large number of word-combinations in even medium-sized text corpora (e.g., the Wikipedia corpus contains $> 10^7$ distinct terms), it is impossible to pre-compute and store the associations for all possible multi-term combinations (e.g., $> 10^{14}$ for 2-way and $> 10^{21}$ for 3-way); instead the techniques described in this paper can be used for fast estimates of the co-occurrences.

**Database query optimization**    Set intersection is a routine operation in databases, employed for example during the evaluation of conjunctive selection conditions in the presence of single-column indexes. Before conducting intersections, a critical task is to (quickly) estimate the sizes of the intermediate results to plan the optimal intersection order [20, 8, 25]. For example, consider the task of intersecting four sets of record identifiers: $A \cap B \cap C \cap D$. Even though the final outcome will be the same, the order of the join operations, e.g., $(A \cap B) \cap (C \cap D)$ or $((A \cap B) \cap C) \cap D$, can significantly affect the performance, in particular if the intermediate results, e.g., $A \cap B \cap C$, become too large for main memory and need to be spilled to disk. A good query plan aims to minimize

the total size of intermediate results. Thus, it is highly desirable to have a mechanism which can estimate join sizes very efficiently, especially for the lower-order (2-way and 3-way) intersections, which could potentially result in much larger intermediate results than higher-order intersections.

**Duplicate Detection in Data Cleaning:** A common task in data cleaning is the identification of duplicates (e.g., duplicate names, organizations, etc.) among a set of items. Now, despite the fact that there is considerable evidence (e.g., [10]) that reliable duplicate-detection should be based on local properties of *groups* of duplicates, most current approaches base their decisions on pairwise similarities between items only. This is in part due to the computational overhead associated with more complex interactions, which our approach may help to overcome.

**Clustering** Most clustering techniques are based on pair-wise distances between the items to be clustered. However, there are a number of natural scenarios where the affinity relations are not pairwise, but rather triadic, tetradic or higher (e.g. [1, 43]). Again, our approach may improve the performance in these scenarios if the distance measures can be expressed in the form of set-overlap.

**Data mining** A lot of work in data mining has focused on efficient candidate pruning in the context of pairwise associations (e.g., [14]), a number of such pruning techniques leverage minwise hashing to prune pairs of items, but in many contexts (e.g., association rules with more than 2 items) multi-way associations are relevant; here, pruning based on pairwise interactions may perform much less well than multi-way pruning.

## 1.1 Ultra-high dimensional data are often binary

For duplicate detection in the context of Web crawling/search, each document can be represented as a set of $w$-shingles ($w$ contiguous words); $w = 5$ or $7$ in several studies [3, 4, 17]. Normally only the abscence/presence (0/1) information is used, as a $w$-shingle rarely occurs more than once in a page if $w \geq 5$. The total number of shingles is commonly set to be $|\Omega| = 2^{64}$; and thus the set intersection corresponds to computing the inner product in binary data vectors of $2^{64}$ dimensions. Interestingly, even when the data are not too high-dimensional (e.g., only thousands), empirical studies [6, 23, 26] achieved good performance using SVM with binary-quantized (text or image) data.

## 1.2 Minwise Hashing and SimHash

Two of the most widely adopted approaches for estimating set intersections are **minwise hashing** [3, 4] and **sign (1-bit) random projections** (also known as **simhash**) [7, 34], which are both special instances of the general techniques proposed in the context of locality-sensitive hashing [7, 24]. These techniques have been successfully applied to many tasks in machine learning, databases, data mining, and information retrieval [18, 36, 11, 22, 16, 39, 28, 41, 27, 5, 2, 37, 7, 24, 21].

**Limitations of random projections** The method of random projections (including simhash) is limited to estimating pairwise similarities. Random projections convert any data distributions to (zero-mean) multivariate normals, whose density functions are determined by the covariance matrix which contains only the pairwise information of the original data. This is a serious limitation.

## 1.3 Prior work on b-Bit Minwise Hashing

Instead of storing each hashed value using 64 bits as in prior studies, e.g., [17], [35] suggested to store only the lowest $b$ bits. [35] demonstrated that using $b = 1$ reduces the storage space at least by a factor of 21.3 (for a given accuracy) compared to $b = 64$, if one is interested in resemblance $\geq 0.5$, the threshold used in prior studies [3, 4]. Moreover, by choosing the value $b$ of bits to be retained, it becomes possible to systematically adjust the degree to which the estimator is "tuned" towards higher similarities as well as the amount of hashing (random permutations) required.

[35] concerned only the pairwise resemblance. To extend it to the multi-way case, we have to solve new and challenging probability problems. Compared to the pairwise case, our new estimator is significantly different. In fact, as we will show later, estimating 3-way resemblance requires $b \geq 2$.

## 1.4 Notation

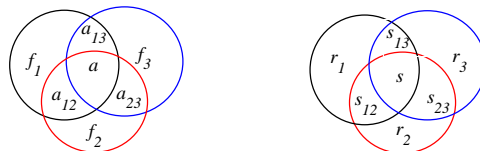

Figure 1: Notation for 2-way and 3-way set intersections.

Fig. 1 describes the notation used in 3-way intersections for three sets $S_1, S_2, S_3 \in \Omega$, $|\Omega| = D$.

- $f_1 = |S_1|$, $f_2 = |S_2|$, $f_3 = |S_3|$.
- $a_{12} = |S_1 \cap S_2|$, $a_{13} = |S_1 \cap S_3|$, $a_{23} = |S_2 \cap S_3|$, $a = a_{123} = |S_1 \cap S_2 \cap S_3|$.
- $r_1 = \frac{f_1}{D}$, $r_2 = \frac{f_2}{D}$, $r_3 = \frac{f_3}{D}$. $s_{12} = \frac{a_{12}}{D}$, $s_{13} = \frac{a_{13}}{D}$, $s_{23} = \frac{a_{23}}{D}$, $s = s_{123} = \frac{a}{D}$.
- $u = r_1 + r_2 + r_3 - s_{12} - s_{13} - s_{23} + s$.

We define three 2-way resemblances ($R_{12}$, $R_{13}$, $R_{23}$) and one 3-way resemblance ($R$) as:

$$R_{12} = \frac{|S_1 \cap S_2|}{|S_1 \cup S_2|}, \quad R_{13} = \frac{|S_1 \cap S_3|}{|S_1 \cup S_3|}, \quad R_{23} = \frac{|S_2 \cap S_3|}{|S_2 \cup S_3|}, \quad R = R_{123} = \frac{|S_1 \cap S_2 \cap S_3|}{|S_1 \cup S_2 \cup S_3|}. \quad (1)$$

which, using our notation, can be expressed in various forms:

$$R_{ij} = \frac{a_{ij}}{f_i + f_j - a_{ij}} = \frac{s_{ij}}{r_i + r_j - s_{ij}}, \; i \neq j, \quad (2)$$

$$R = \frac{a}{f_1 + f_2 + f_3 - a_{12} - a_{23} - a_{13} + a} = \frac{s}{r_1 + r_2 + r_3 - s_{12} - s_{23} - s_{13} + s} = \frac{s}{u}. \quad (3)$$

Note that, instead of $a_{123}$, $s_{123}$, $R_{123}$, we simply use $a$, $s$, $R$. When the set sizes, $f_i = |S_i|$, can be assumed to be known, we can compute resemblances from intersections and vice versa:

$$a_{ij} = \frac{R_{ij}}{1 + R_{ij}}(f_i + f_j), \qquad a = \frac{R}{1 - R}(f_1 + f_2 + f_3 - a_{12} - a_{13} - a_{23}).$$

Thus, estimating resemblances and estimating intersection sizes are two closely related problems.

## 1.5 Our Main Contributions

- We derive the basic probability formula for estimating 3-way resemblance using $b$-bit hashing. The derivation turns out to be significantly much more complex than the 2-way case. This basic probability formula naturally leads to a (complicated) estimator of resemblance.

- We leverage the observation that many real applications involve sparse data (i.e., $r_i = \frac{f_i}{D} \approx 0$, but $f_i/f_j = r_i/r_j$ may be still significant) to develop a much simplified estimator, which is desired in practical applications. This assumption of $f_i/D \to 0$ significantly simplifies the estimator and frees us from having to know the cardinalities $f_i$.

- We analyze the theoretical variance of the simplified estimator and compare it with the original minwise hashing method (using 64 bits). Our theoretical analysis shows that $b$-bit minwise hashing can normally achieve a 10 to 25-fold improvement in storage space (for a given estimator accuracy of the 3-way resemblance) when the set similarities are not extremely low (e.g., when the 3-way resemblance $> 0.02$). These results are particularly important for applications in which only detecting high resemblance/overlap is relevant, such as many data cleaning scenarios or duplicate detection.

The recommended procedure for estimating 3-way resemblances (in sparse data) is shown as Alg. 1.

---

**Algorithm 1** The $b$-bit minwise hashing algorithm, applied to estimating 3-way resemblances in a collection of $N$ sets. This procedure is suitable for sparse data, i.e., $r_i = f_i/D \approx 0$.

---
**Input:** Sets $S_n \in \Omega = \{0, 1, ..., D - 1\}$, $n = 1$ to $N$.
**Pre-processing phrase:**
1) Generate $k$ random permutations $\pi_j : \Omega \to \Omega$, $j = 1$ to $k$.
2) For each set $S_n$ and permutation $\pi_j$, store the lowest $b$ bits of $\min(\pi_j(S_n))$, denoted by $e_{n,t,\pi_j}$, $t = 1$ to $b$.
**Estimation phrase:** (Use three sets $S_1$, $S_2$, and $S_3$ as an example.)
1) Compute $\hat{P}_{12,b} = \frac{1}{k}\sum_{j=1}^{k}\left\{\prod_{t=1}^{b}1\{e_{1,t,\pi_j} = e_{2,t,\pi_j}\}\right\}$. Similarly, compute $\hat{P}_{13,b}$ and $\hat{P}_{23,b}$.
2) Compute $\hat{P}_b = \frac{1}{k}\sum_{j=1}^{k}\left\{\prod_{t=1}^{b}1\{e_{1,t,\pi_j} = e_{2,t,\pi_j} = e_{3,t,\pi_j}\}\right\}$.
3) Estimate $R$ by $\hat{R}_b = \frac{4^b\hat{P}_b - 2^b\left(\hat{P}_{12,b} + \hat{P}_{13,b} + \hat{P}_{23,b}\right) + 2}{(2^b - 1)(2^b - 2)}$.
4) If needed, the 2-way resemblances $R_{ij,b}$ can be estimated as $\hat{R}_{ij,b} = \frac{2^b\hat{P}_{ij,b} - 1}{2^b - 1}$.

---

## 2 The Precise Theoretical Probability Analysis

Minwise hashing applies $k$ random permutations $\pi_j : \Omega \longrightarrow \Omega$, $\Omega = \{0, 1, ..., D - 1\}$, and then estimates $R_{12}$ (and similarly other 2-way resemblances) using the following probability:

$$\mathbf{Pr}\left(\min(\pi_j(S_1)) = \min(\pi_j(S_2))\right) = \frac{|S_1 \cap S_2|}{|S_1 \cup S_2|} = R_{12}. \tag{4}$$

This method naturally extends to estimating 3-way resemblances for three sets $S_1, S_2, S_3 \in \Omega$:

$$\mathbf{Pr}\left(\min(\pi_j(S_1)) = \min(\pi_j(S_2)) = \min(\pi_j(S_3))\right) = \frac{|S_1 \cap S_2 \cap S_3|}{|S_1 \cup S_2 \cup S_3|} = R. \tag{5}$$

To describe $b$-bit hashing, we define the minimum values under $\pi$ and their lowest $b$ bits to be:

$$z_i = \min\left(\pi\left(S_i\right)\right), \qquad e_{i,t} = t\text{-th lowest bit of } z_i.$$

To estimate $R$, we need to computes the empirical estimates of the probabilities $P_{ij,b}$ and $P_b$, where

$$P_{ij,b} = \mathbf{Pr}\left(\prod_{t=1}^{b} 1\{e_{i,t} = e_{j,t}\} = 1\right), \qquad P_b = P_{123,b} = \mathbf{Pr}\left(\prod_{t=1}^{b} 1\{e_{1,t} = e_{2,t} = e_{3,t}\} = 1\right).$$

The main theoretical task is to derive $P_b$. The prior work[35] already derived $P_{ij,b}$; see Appendix A. To simplify the algebra, we assume that $D$ is large, which is virtually always satisfied in practice.

**Theorem 1** *Assume $D$ is large.*

$$P_b = \mathbf{Pr}\left(\prod_{i=1}^{b} 1\{e_{1,i} = e_{2,i} = e_{3,i}\} = 1\right) = \frac{Z}{u} + R = \frac{Z + s}{u}, \tag{6}$$

*where $u = r_1 + r_2 + r_3 - s_{12} - s_{13} - s_{23} + s$, and*

$$Z = (s_{12} - s)A_{3,b} + \frac{(r_3 - s_{13} - s_{23} + s)}{r_1 + r_2 - s_{12}}s_{12}G_{12,b} + (s_{13} - s)A_{2,b} + \frac{(r_2 - s_{12} - s_{23} + s)}{r_1 + r_3 - s_{13}}s_{13}G_{13,b}$$

$$+ (s_{23} - s)A_{1,b} + \frac{(r_1 - s_{12} - s_{13} + s)}{r_2 + r_3 - s_{23}}s_{23}G_{23,b} + \left[(r_2 - s_{23})A_{3,b} + (r_3 - s_{23})A_{2,b}\right]\frac{(r_1 - s_{12} - s_{13} + s)}{r_2 + r_3 - s_{23}}G_{23,b}$$

$$+ \left[(r_1 - s_{13})A_{3,b} + (r_3 - s_{13})A_{1,b}\right]\frac{(r_2 - s_{12} - s_{23} + s)}{r_1 + r_3 - s_{13}}G_{13,b}$$

$$+ \left[(r_1 - s_{12})A_{2,b} + (r_2 - s_{12})A_{1,b}\right]\frac{(r_3 - s_{13} - s_{23} + s)}{r_1 + r_2 - s_{12}}G_{12,b},$$

$$A_{j,b} = \frac{r_j(1 - r_j)^{2^b - 1}}{1 - (1 - r_j)^{2^b}}, \qquad G_{ij,b} = \frac{(r_i + r_j - s_{ij})(1 - r_i - r_j + s_{ij})^{2^b - 1}}{1 - (1 - r_i - r_j + s_{ij})^{2^b}}, \quad i, j \in \{1, 2, 3\}, \ i \neq j.$$

Theorem 1 naturally suggests an iterative estimation procedure, by writing Eq. (6) as $s = P_b u - Z$.

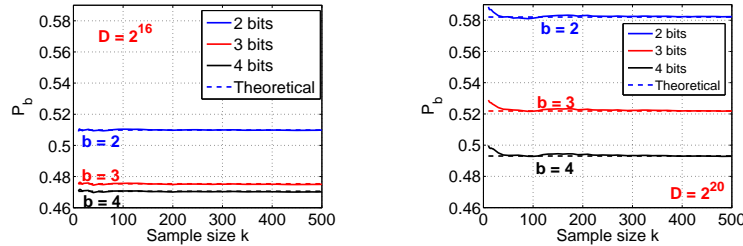

Figure 2: $P_b$, for verifying the probability formula in Theorem 1. The empirical estimates and the theoretical predictions essentially overlap regardless of the sparsity measure $r_i = f_i/D$.

**A Simulation Study**    For the purpose of verifying Theorem 1, we use three sets corresponding to the occurrences of three common words ("OF", "AND", and "OR") in a chunk of real world Web crawl data. Each (word) set is a set of document (Web page) IDs which contained that word at least once. The three sets are not too sparse and $D = 2^{16}$ suffices to represent their elements. The $r_i = \frac{f_i}{D}$ values are 0.5697, 0.5537, and 0.3564, respectively. The true 3-way resemblance is $R = 0.47$.

We can also increase $D$ by mapping these sets into a larger space using a random mapping, with $D = 2^{16}, 2^{18}, 2^{20}$, or $2^{22}$. When $D = 2^{22}$, the $r_i$ values are 0.0089, 0.0087, 0.0056.

Fig. 2 presents the empirical estimates of the probability $P_b$, together with the theoretical predictions by Theorem 1. The empirical estimates essentially overlap the theoretical predictions. Even though the proof assumes $D \to \infty$, $D$ does not have to be too large for Theorem 1 to be accurate.

## 3 The Much Simplified Estimator for Sparse Data

The basic probability formula (Theorem 1) we derive could be too complicated for practical use. To obtain a simpler formula, we leverage the observation that in practice we often have $r_i = \frac{f_i}{D} \approx 0$, even though both $f_i$ and $D$ can be very large. For example, consider web duplicate detection [17]. Here, $D = 2^{64}$, which means that even for a web page with $f_i = 2^{54}$ shingles (corresponding to the text of a small novel), we still have $\frac{f_i}{D} \approx 0.001$. Note that, even when $r_i \to 0$, the ratios, e.g., $\frac{r_2}{r_1}$, can be still large. Recall the resemblances (2) and (3) are only determined by these ratios.

We analyzed the distribution of $\frac{f_i}{D}$ using two real-life datasets: the UCI dataset containing $3 \times 10^5$ NYTimes articles; and a Microsoft proprietary dataset with $10^6$ news articles [19]. For the UCI-NYTimes dataset, each document was already processed as a set of single words. For the anonymous dataset, we report results using three different representations: single words (1-shingle), 2-shingles (two contiguous words), and 3-shingles. Table 1 reports the summary statistics of the $\frac{f_i}{D}$ values.

Table 1: Summary statistics of the $\frac{f_i}{D}$ values in two datasets

| Data | Median | Mean | Std. |
|---|---|---|---|
| $3 \times 10^5$ UCI-NYTimes articles | 0.0021 | 0.0022 | 0.0011 |
| $10^6$ Microsoft articles (1-shingle) | 0.00027 | 0.00032 | 0.00023 |
| $10^6$ Microsoft articles (2-shingle) | 0.00003 | 0.00004 | 0.00005 |
| $10^6$ Microsoft articles (3-shingle) | 0.00002 | 0.00002 | 0.00002 |

For truly large-scale applications, prior studies [3, 4, 17] commonly used 5-shingles. This means that real world data may be significantly more sparse than the values reported in Table 1.

### 3.1 The Simplified Probability Formula and the Practical Estimator

**Theorem 2** *Assume $D$ is large. Let $T = R_{12} + R_{13} + R_{23}$. As $r_1, r_2, r_3 \to 0$,*

$$P_b = \mathbf{Pr}\left(\prod_{i=1}^{b} 1\{e_{1,i} = e_{2,i} = e_{3,i}\} = 1\right) = \frac{1}{4^b}\left\{(2^b - 1)(2^b - 2)R + (2^b - 1)T + 1\right\}. \quad (7)$$

Interestingly, if $b = 1$, then $P_1 = \frac{1}{4}(1 + T)$, i.e., no information about the 3-way resemblance $R$ is contained. Hence, it is necessary to use $b \geq 2$ to estimate 3-way similarities.

Alg. 1 uses $\hat{P}_b$ and $\hat{P}_{ij,b}$ to respectively denote the empirical estimates of the theoretical probabilities $P_b$ and $P_{ij,b}$. Assuming $r_1, r_2, r_3 \to 0$, the proposed estimator of $R$, denoted by $\hat{R}_b$, is

$$\hat{R}_b = \frac{4^b \hat{P}_b - 2^b \left(\hat{P}_{12,b} + \hat{P}_{13,b} + \hat{P}_{23,b}\right) + 2}{(2^b - 1)(2^b - 2)}. \quad (8)$$

**Theorem 3** *Assume $D$ is large and $r_1, r_2, r_3 \to 0$. Then $\hat{R}_b$ in (8) is unbiased with the variance*

$$Var\left(\hat{R}_b\right) = \frac{1}{k}\frac{1}{(2^b - 1)(2^b - 2)}\left\{1 + (2^b - 3)T + \left(4^b - 6 \times 2^b + 10\right)R - (2^b - 1)(2^b - 2)R^2\right\}. \quad (9)$$

It is interesting to examine several special cases:

- $b = 1$: $Var(\hat{R}_1) = \infty$, i.e., one must use $b \geq 2$.
- $b = 2$: $Var(\hat{R}_2) = \frac{1}{6k}\left(1 + T + 2R - 6R^2\right)$.
- $b = \infty$: $Var(\hat{R}_\infty) = \frac{1}{k}R(1 - R) = Var(\hat{R}_M)$. $\hat{R}_M$ is the original minwise hashing estimator for 3-way resemblance. In principle, the estimator $\hat{R}_M$ requires an infinite precision (i.e., $b = \infty$). Numerically, $Var(\hat{R}_M)$ and $Var(\hat{R}_{64})$ are indistinguishable.

## 3.2 Simulations for Validating Theorem 3

We now present a simulation study for verifying Theorem 3, using the same three sets used in Fig. 2.

Fig. 3 presents the resulting empirical biases: $E(\hat{R}_b) - R_b$. Fig. 4 presents the empirical mean square errors (MSE = bias$^2$+variance) together with the theoretical variances $Var(\hat{R}_b)$ in Theorem 3.

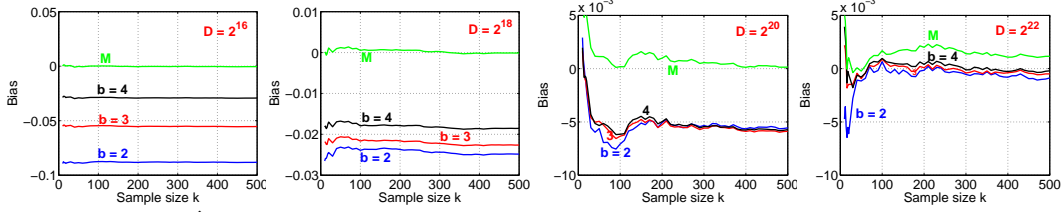

Figure 3: Bias of $\hat{R}_b$ (8). We used 3 (word) sets: "OF", "AND", and "OR" and four $D$ values: $2^{16}$, $2^{18}$, $2^{20}$, and $2^{22}$. We conducted experiments using $b = 2$, 3, and 4 as well as the original minwise hashing (denoted by "M"). The plots verify that as $r_i$ decreases (to zero), the biases vanish. Note that the set sizes $f_i$ remain the same, but the relative values $r_i = \frac{f_i}{D}$ decrease as $D$ increases.

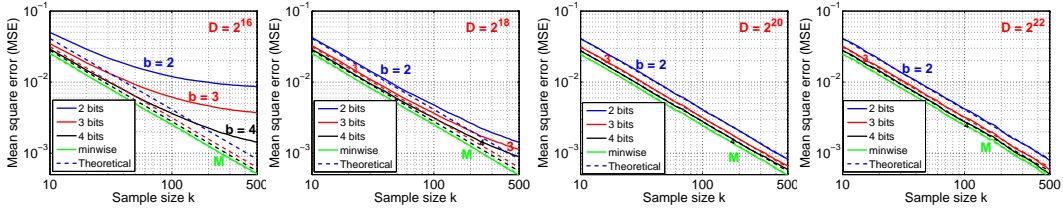

Figure 4: MSE of $\hat{R}_b$ (8). The solid curves are the empirical MSEs (=var+bias$^2$) and the dashed lines are the theoretical variances (9), under the assumption of $r_i \to 0$. Ideally, we would like to see the solid and dashed lines overlap. When $D = 2^{20}$ and $D = 2^{22}$, even though the $r_i$ values are not too small, the solid and dashed lines almost overlap. Note that, at the same sample size $k$, we always have $Var(\hat{R}_2) > Var(\hat{R}_3) > Var(\hat{R}_4) > Var(\hat{R}_M)$, where $\hat{R}_M$ is the original minwise hashing estimator. We can see that, $Var(\hat{R}_3)$ and $Var(\hat{R}_4)$ are very close to $Var(\hat{R}_M)$.

We can summarize the results in Fig. 3 and Fig. 4 as follows:

- When the $r_i = \frac{f_i}{D}$ values are large (e.g., $r_i \approx 0.5$ when $D = 2^{16}$), the estimates using (8) can be noticeably biased. The estimation biases diminish as the $r_i$ values decrease. In fact, even when the $r_i$ values are not small (e.g., $r_i \approx 0.05$ when $D = 2^{20}$), the biases are already very small (roughly 0.005 when $D = 2^{20}$).
- The variance formula (9) becomes accurate when the $r_i$ values are not too large. For example, when $D = 2^{18}$ ($r_i \approx 0.1$), the empirical MSEs largely overlap the theoretical variances which assumed $r_i \to 0$, unless the sample size $k$ is large. When $D = 2^{20}$ (and $D = 2^{22}$), the empirical MSEs and theoretical variances overlap.
- For real applications, as we expect $D$ will be very large (e.g., $2^{64}$) and the $r_i$ values ($f_i/D$) will be very small, our proposed simple estimator (8) will be very useful in practice, because it becomes unbiased and the variance can be reliably predicted by (9).

## 4 Improving Estimates for Dense Data Using Theorem 1

While we believe the simple estimator in (8) and Alg. 1 should suffice in most applications, we demonstrate here that the sparsity assumption of $r_i \to 0$ is not essential if one is willing to use the more sophisticated estimation procedure provided by Theorem 1.

By Eq. (6), $s = P_b u - Z$, where $Z$ contains $s$, $s_{ij}$, $r_i$ etc. We first estimate $s_{ij}$ (from the estimated $R_{ij}$) using the precise formula for the two-way case; see Appendix A. We then iteratively solve for $s$ using the initial guess provided by the estimator $\hat{R}_b$ in (8). Usually a few iterations suffice.

Fig. 5 reports the bias (left most panel, only for $D = 2^{16}$) and MSE, corresponding to Fig. 3 and Fig. 4. In Fig. 5, the solid curves are obtained using the precise estimation procedure by Theorem 1. The dashed curves are the estimates using the simplified estimator $\hat{R}_b$ which assumes $r_i \to 0$.

Even when the data are not sparse, the precise estimation procedure provides unbiased estimates as verified by the leftmost panel of Fig. 5. Using the precise procedure results in noticeably more accurate estimates in non-sparse data, as verified by the second panel of Fig. 5. However, as long as the data are reasonably sparse (the right two panels), the simple estimator $\hat{R}_b$ in (8) is accurate.

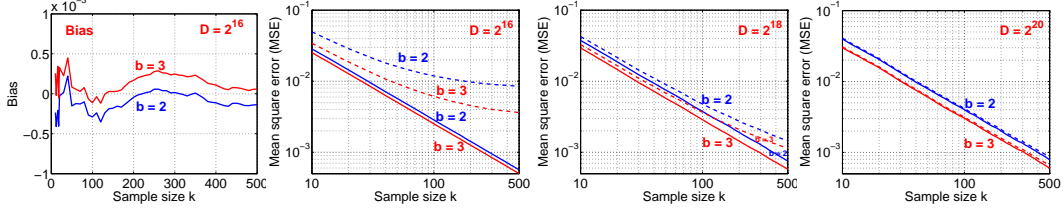

Figure 5: The bias (leftmost panel) and MSE of the precise estimation procedure, using the same data used in Fig. 3 and Fig. 4. The dashed curves correspond to the estimates using the simplified estimator $\hat{R}_b$ in (8) which assumes $r_i \to 0$.

# 5 Quantifying the Improvements Using $b$-Bit Hashing

This section is devoted to analyzing the improvements of $b$-bit minwise hashing, compared to using 64 bits for each hashed value. Throughout the paper, we use the terms "**sample**" and "**sample size**" (denoted by $k$). The original minwise hashing stores each "sample" using 64 bits (as in [17]). For $b$-bit minwise hashing, we store each "sample" using $b$ bits only. Note that $Var(\hat{R}_{64})$ and $Var(\hat{R}_M)$ (the variance of the original minwise hashing) are numerically indistinguishable.

As we decrease $b$, the space needed for each sample will be smaller; the estimation variance at the same sample size $k$, however, will increase. This variance-space trade-off can be quantified by $B(b) = b \times \mathrm{Var}\left(\hat{R}_b\right) \times k$, which is called the ***storage factor***. Lower $B(b)$ is more desirable. The ratio $\frac{B(64)}{B(b)}$ precisely characterizes the improvements of $b$-bit hashing compared to using 64 bits.

Fig. 6 confirms the substantial improvements of $b$-bit hashing over the original minwise hashing using 64 bits. The improvements in terms of the storage space are usually 10 (or 15) to 25-fold when the sets are reasonably similar (i.e., when the 3-way resemblance $> 0.1$). When the three sets are very similar (e.g., the top left panel), the improvement will be even 25 to 30-fold.

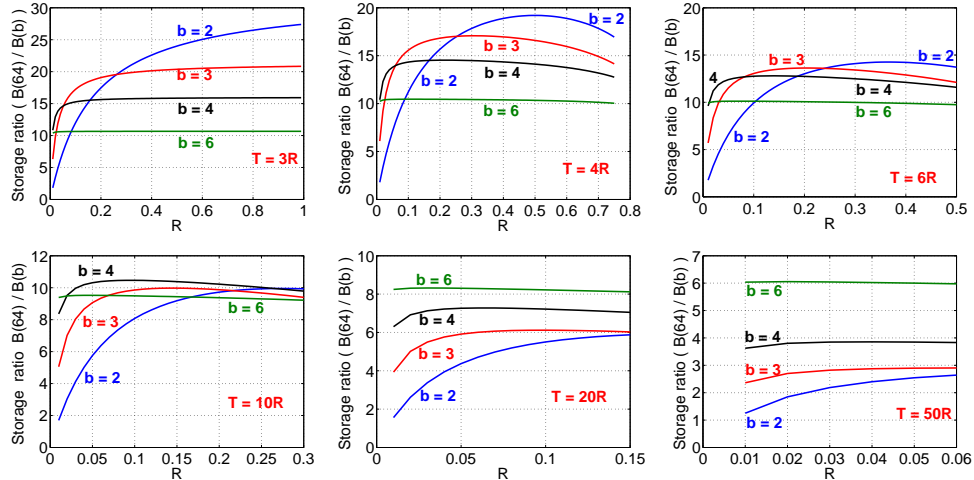

Figure 6: $\frac{B(64)}{B(b)}$, the relative storage improvement of using $b = 2, 3, 4, 6$ bits, compared to using $64$ bits. Since the variance (9) contains both $R$ and $T = R_{12} + R_{13} + R_{23}$, we compare variances using different $T/R$ ratios. As $3R \leq T$ always, we let $T = \alpha R$, for some $\alpha \geq 3$. Since $T \leq 3$, we know $R \leq 3/\alpha$. Practical applications are often interested in cases with reasonably large $R$ values.

## 6 Evaluation of Accuracy

We conducted a duplicate detection experiment on a public (UCI) collection of 300,000 NYTimes news articles. The task is to identify 3-groups with 3-way resemblance $R$ exceeding a threshold $\mathbf{R_0}$. We used a subset of the data; the total number of 3-groups is about one billion. We experimented with $b = 2, 4$ and the original minwise hashing. Fig. 7 presents the precision curves for a representative set of thresholds $R_0$'s. Just like in [35], the recall curves are not shown because they could not differentiate estimators. These curves confirm the significant improvement of using $b$-bit minwise hashing when the threshold $R_0$ is quite high (e.g., 0.3). In fact, when $R_0 = 0.3$, using $b = 4$ resulted in similar precisions as using the original minwise hashing (i.e., a 64/4=16-fold reduction in storage). Even when $R_0 = 0.1$, using $b = 4$ can still achieve similar precisions as using the original minwise hashing by only slightly increasing the sample size $k$.

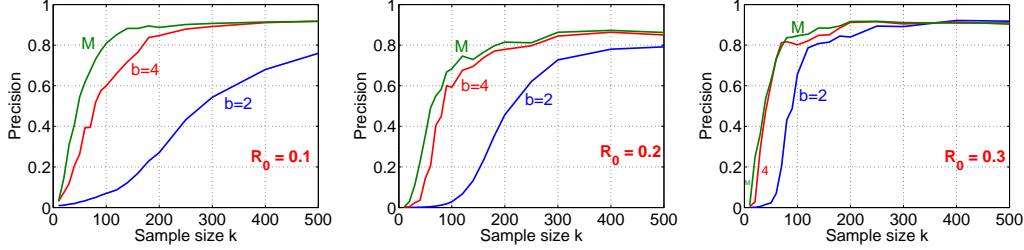

Figure 7: Precision curves on the UCI collection of news data. The task is to retrieve news article 3-groups with resemblance $R \geq R_0$. For example, consider $R_0 = 0.2$. To achieve a precision of at least 0.8, 2-bit hashing and 4-bit hashing require about $k = 500$ samples and $k = 260$ samples respectively, while the original minwise hashing (denoted by $M$) requires about 170 samples.

## 7 Conclusion

Computing set similarities is fundamental in many applications. In machine learning, high-dimensional binary data are common and are equivalent to sets. This study is devoted to simultaneously estimating 2-way and 3-way similarities using $b$-bit minwise hashing. Compared to the prior work on estimating 2-way resemblance [35], the extension to 3-way is important for many application scenarios (as described in Sec. 1) and is technically non-trivial.

For estimating 3-way resemblance, our analysis shows that $b$-bit minwise hashing can normally achieve a 10 to 25-fold improvement in the storage space required for a given estimator accuracy, when the set similarities are not extremely low (e.g., 3-way resemblance > 0.02). Many applications such as data cleaning and de-duplication are mainly concerned with relatively high set similarities.

For many practical applications, the reductions in storage directly translate to improvements in processing speed as well, especially when memory latency is the main bottleneck, which, with the advent of many-core processors, is more and more common.

**Future work**: We are interested in developing a $b$-bit version for *Conditional Random Sampling (CRS)* [31, 32, 33], which requires only one permutation (instead of $k$ permutations) and naturally extends to non-binary data. CRS is also provably more accurate than minwise hashing for binary data. However, the analysis for developing the $b$-bit version of CRS appears to be very difficult.

## A Review of $b$-Bit Minwise Hashing for 2-Way Resemblance

**Theorem 4 ([35])** *Assume $D$ is large.*

$$P_{12,b} = \mathbf{Pr}\left(\prod_{i=1}^{b} 1\{e_{1,i} = e_{2,i}\} = 1\right) = C_{1,b} + (1 - C_{2,b})R_{12}$$

*where* 
$$C_{1,b} = A_{1,b}\frac{r_2}{r_1 + r_2} + A_{2,b}\frac{r_1}{r_1 + r_2}, \quad C_{2,b} = A_{1,b}\frac{r_1}{r_1 + r_2} + A_{2,b}\frac{r_2}{r_1 + r_2},$$

$$A_{1,b} = \frac{r_1[1 - r_1]^{2^b - 1}}{1 - [1 - r_1]^{2^b}}, \qquad A_{2,b} = \frac{r_2[1 - r_2]^{2^b - 1}}{1 - [1 - r_2]^{2^b}}.$$

If $r_1, r_2 \to 0$, $P_{12,b} = \frac{1 + (2^b - 1)R_{12}}{2^b}$ and one can estimate $R_{12}$ by $\frac{2^b \hat{P}_{12,b} - 1}{2^b - 1}$, where $\hat{P}_{12,b}$ is the empirical observation of $P_{12,b}$. If $r_1, r_2$ are not small, $R_{12}$ is estimated by $(\hat{P}_{12,b} - C_{1,b})/(1 - C_{2,b})$.

## Footnotes

[1]This work is supported by NSF (DMS-0808864), ONR (YIP-N000140910911) and Microsoft.

# References

[1] S. Agarwal, J. Lim, L. Zelnik-Manor, P. Perona, D. Kriegman, and S. Belongie. Beyond pairwise clustering. In *CVPR*, 2005.

[2] M. Bendersky and W. B. Croft. Finding text reuse on the web. In *WSDM*, pages 262–271, Barcelona, Spain, 2009.

[3] A. Z. Broder. On the resemblance and containment of documents. In *the Compression and Complexity of Sequences*, pages 21–29, Positano, Italy, 1997.

[4] A. Z. Broder, S. C. Glassman, M. S. Manasse, and G. Zweig. Syntactic clustering of the web. In *WWW*, pages 1157 – 1166, Santa Clara, CA, 1997.

[5] G. Buehrer and K. Chellapilla. A scalable pattern mining approach to web graph compression with communities. In *WSDM*, pages 95–106, Stanford, CA, 2008.

[6] O. Chapelle, P. Haffner, and V. N. Vapnik. Support vector machines for histogram-based image classification. 10(5):1055–1064, 1999.

[7] M. S. Charikar. Similarity estimation techniques from rounding algorithms. In *STOC*, pages 380–388, Montreal, Quebec, Canada, 2002.

[8] S. Chaudhuri. An Overview of Query Optimization in Relational Systems. In *PODS*, pages 34–43, 1998.

[9] S. Chaudhuri, V. Ganti, and R. Kaushik. A primitive operatior for similarity joins in data cleaning. In *ICDE*, 2006.

[10] S. Chaudhuri, V. Ganti, and R. Motwani. Robust identification of fuzzy duplicates. In *ICDE*, pages 865–876, Tokyo, Japan, 2005.

[11] F. Chierichetti, R. Kumar, S. Lattanzi, M. Mitzenmacher, A. Panconesi, and P. Raghavan. On compressing social networks. In *KDD*, pages 219–228, Paris, France, 2009.

[12] K. Church. Approximate lexicography and web search. *International Journal of Lexicography*, 21(3):325–336, 2008.

[13] K. Church and P. Hanks. Word association norms, mutual information and lexicography. *Computational Linguistics*, 16(1):22–29, 1991.

[14] E. Cohen, M. Datar, S. Fujiwara, A. Gionis, P. Indyk, R. Motwani, J. D. Ullman, and C. Yang. Finding interesting associations without support pruning. *IEEE Trans. on Knowl. and Data Eng.*, 13(1), 2001.

[15] F. Diaz. Integration of News Content into Web Results. In *WSDM*, 2009.

[16] Y. Dourisboure, F. Geraci, and M. Pellegrini. Extraction and classification of dense implicit communities in the web graph. *ACM Trans. Web*, 3(2):1–36, 2009.

[17] D. Fetterly, M. Manasse, M. Najork, and J. L. Wiener. A large-scale study of the evolution of web pages. In *WWW*, pages 669–678, Budapest, Hungary, 2003.

[18] G. Forman, K. Eshghi, and J. Suermondt. Efficient detection of large-scale redundancy in enterprise file systems. *SIGOPS Oper. Syst. Rev.*, 43(1):84–91, 2009.

[19] M. Gamon, S. Basu, D. Belenko, D. Fisher, M. Hurst, and A. C. König. Blews: Using blogs to provide context for news articles. In *AAAI Conference on Weblogs and Social Media*, 2008.

[20] H. Garcia-Molina, J. D. Ullman, and J. Widom. *Database Systems: the Complete Book*. Prentice Hall, New York, NY, 2002.

[21] A. Gionis, D. Gunopulos, and N. Koudas. Efficient and tunable similar set retrieval. In *SIGMOD*, pages 247–258, CA, 2001.

[22] S. Gollapudi and A. Sharma. An axiomatic approach for result diversification. In *WWW*, pages 381–390, Madrid, Spain, 2009.

[23] M. Hein and O. Bousquet. Hilbertian metrics and positive definite kernels on probability measures. In *AISTATS*, pages 136–143, Barbados, 2005.

[24] P. Indyk and R. Motwani. Approximate nearest neighbors: Towards removing the curse of dimensionality. In *STOC*, pages 604–613, Dallas, TX, 1998.

[25] Y. E. Ioannidis. The history of histograms (abridged). In *VLDB*, 2003.

[26] Y. Jiang, C. Ngo, and J. Yang. Towards optimal bag-of-features for object categorization and semantic video retrieval. In *CIVR*, pages 494–501, Amsterdam, Netherlands, 2007.

[27] N. Jindal and B. Liu. Opinion spam and analysis. In *WSDM*, pages 219–230, Palo Alto, California, USA, 2008.

[28] K. Kalpakis and S. Tang. Collaborative data gathering in wireless sensor networks using measurement co-occurrence. *Computer Communications*, 31(10):1979–1992, 2008.

[29] A. C. König, M. Gamon, and Q. Wu. Click-Through Prediction for News Queries. In *SIGIR*, 2009.

[30] H. Lee, R. T. Ng, and K. Shim. Power-law based estimation of set similarity join size. In *PVLDB*, 2009.

[31] P. Li and K. W. Church. A sketch algorithm for estimating two-way and multi-way associations. *Computational Linguistics*, 33(3):305–354, 2007 (Preliminary results appeared in HLT/EMNLP 2005).

[32] P. Li, K. W. Church, and T. J. Hastie. Conditional random sampling: A sketch-based sampling technique for sparse data. In *NIPS*, pages 873–880, Vancouver, BC, Canada, 2006.

[33] P. Li, K. W. Church, and T. J. Hastie. One sketch for all: Theory and applications of conditional random sampling. In *NIPS*, Vancouver, BC, Canada, 2008.

[34] P. Li, T. J. Hastie, and K. W. Church. Improving random projections using marginal information. In *COLT*, pages 635–649, Pittsburgh, PA, 2006.

[35] P. Li and A. C. König. b-bit minwise hashing. In *WWW*, pages 671–680, Raleigh, NC, 2010.

[36] Ludmila, K. Eshghi, C. B. M. III, J. Tucek, and A. Veitch. Probabilistic frequent itemset mining in uncertain databases. In *KDD*, pages 1087–1096, Paris, France, 2009.

[37] G. S. Manku, A. Jain, and A. D. Sarma. Detecting Near-Duplicates for Web-Crawling. In *WWW*, Banff, Alberta, Canada, 2007.

[38] C. D. Manning and H. Schutze. *Foundations of Statistical Natural Language Processing*. The MIT Press, Cambridge, MA, 1999.

[39] M. Najork, S. Gollapudi, and R. Panigrahy. Less is more: sampling the neighborhood graph makes salsa better and faster. In *WSDM*, pages 242–251, Barcelona, Spain, 2009.

[40] S. Sarawagi and A. Kirpal. Efficient set joins on similarity predicates. In *SIGMOD*, pages 743–754, 2004.

[41] T. Urvoy, E. Chauveau, P. Filoche, and T. Lavergne. Tracking web spam with html style similarities. *ACM Trans. Web*, 2(1):1–28, 2008.

[42] X. Wang and C. Zhai. Mining term association patterns from search logs for effective query reformulation. In *CIKM*, pages 479–488, Napa Valley, California, USA, 2008.

[43] D. Zhou, J. Huang, and B. Schölkopf. Beyond pairwise classification and clustering using hypergraphs. 2006.
